# Intrusion Detection with Neural Networks

**Jake Ryan***
Department of Computer Sciences
The University of Texas at Austin
Austin, TX 78712
raven@cs.utexas.edu

**Meng-Jang Lin**
Department of Electrical and Computer Engineering
The University of Texas at Austin
Austin, TX 78712
mj@orac.ece.utexas.edu

**Risto Miikkulainen**
Department of Computer Sciences
The University of Texas at Austin
Austin, TX 78712
risto@cs.utexas.edu

## Abstract

With the rapid expansion of computer networks during the past few years, security has become a crucial issue for modern computer systems. A good way to detect illegitimate use is through monitoring unusual user activity. Methods of intrusion detection based on hand-coded rule sets or predicting commands on-line are laborous to build or not very reliable. This paper proposes a new way of applying neural networks to detect intrusions. We believe that a user leaves a 'print' when using the system; a neural network can be used to learn this print and identify each user much like detectives use thumbprints to place people at crime scenes. If a user's behavior does not match his/her print, the system administrator can be alerted of a possible security breech. A backpropagation neural network called NNID (Neural Network Intrusion Detector) was trained in the identification task and tested experimentally on a system of 10 users. The system was 96% accurate in detecting unusual activity, with 7% false alarm rate. These results suggest that learning user profiles is an effective way for detecting intrusions.

## 1  INTRODUCTION

Intrusion detection schemes can be classified into two categories: misuse and anomaly intrusion detection. Misuse refers to known attacks that exploit the known vulnerabilities of the system. Anomaly means unusual activity in general that could indicate an intrusion.

If the observed activity of a user deviates from the expected behavior, an anomaly is said to occur.

Misuse detection can be very powerful on those attacks that have been programmed in to the detection system. However, it is not possible to anticipate all the different attacks that could occur, and even the attempt is laborous. Some kind of anomaly detection is ultimately necessary. One problem with anomaly detection is that it is likely to raise many false alarms. Unusual but legitimate use may sometimes be considered anomalous. The challenge is to develop a model of legitimate behavior that would accept novel legitimate use.

It is difficult to build such a model for the same reason that it is hard to build a comprehensive misuse detection system: it is not possible to anticipate all possible variations of such behavior. The task can be made tractable in three ways: (1) Instead of general legitimate use, the behavior of individual users in a particular system can be modeled. The task of characterizing regular patterns in the behavior of an individual user is an easier task than trying to do it for all users simultaneously. (2) The patterns of behavior can be learned for examples of legitimate use, instead of having to describe them by hand-coding possible behaviors. (3) Detecting an intrusion real-time, as the user is typing commands, is very difficult because the order of commands can vary a lot. In many cases it is enough to recognize that the distribution of commands over the entire login session, or even the entire day, differs from the usual.

The system presented in this paper, NNID (Neural Network Intrusion Detector), is based on these three ideas. NNID is a backpropagation neural network trained to identify users based on what commands they use during a day. The system administrator runs NNID at the end of each day to see if the users' sessions match their normal pattern. If not, an investigation can be launched. The NNID model is implemented in a UNIX environment and consists of keeping logs of the commands executed, forming command histograms for each user, and learning the users' profiles from these histograms. NNID provides an elegant solution to off-line monitoring utilizing these user profiles. In a system of 10 users, NNID was 96% accurate in detecting anomalous behavior (i.e. random usage patterns), with a false alarm rate of 7%. These results show that a learning offline monitoring system such as NNID can achieve better performance than systems that attempt to detect anomalies on-line in the command sequences, and with computationally much less effort.

The rest of the paper outlines other approaches to intrusion detection and motivates the NNID approach in more detail (sections 2 and 3), presents the implementation and an evaluation on a real-world computer system (sections 4 and 5), and outlines some open issues and avenues for future work (section 6).

## 2   INTRUSION DETECTION SYSTEMS

Many misuse and anomaly intrusion detection systems (IDSs) are based on the general model proposed by Denning (1987). This model is independent of the platform, system vulnerability, and type of intrusion. It maintains a set of historical profiles for users, matches an audit record with the appropriate profile, updates the profile whenever necessary, and reports any anomalies detected. Another component, a rule set, is used for detecting misuse.

Actual systems implement the general model with different techniques (see Frank 1994; Mukherjee et al. 1994, for an overview). Often statistical methods are used to measure how anomalous the behavior is, that is, how different e.g. the commands used are from normal behavior. Such approaches require that the distribution of subjects' behavior is known. The behavior can be represented as a rule-based model (Garvey and Lunt 1991), in terms of predictive pattern generation (Teng et al. 1990), or using state transition analysis (Porras

et al. 1995). Pattern matching techniques are then used to determine whether the sequence of events is part of normal behavior, constitutes an anomaly, or fits the description of a known attack.

IDSs also differ in whether they are on-line or off-line. Off-line IDSs are run periodically and they detect intrusions after-the-fact based on system logs. On-line systems are designed to detect intrusions while they are happening, thereby allowing for quicker intervention. On-line IDSs are computationally very expensive because they require continuous monitoring. Decisions need to be made quickly with less data and therefore they are not as reliable.

Several IDSs that employ neural networks for on-line intrusion detection have been proposed (Debar et al. 1992; Fox et al. 1990). These systems learn to predict the next command based on a sequence of previous commands by a specific user. Through a shifting window, the network receives the $w$ most recent commands as its input. The network is recurrent, that is, part of the output is fed back as the input for the next step; thus, the network is constantly observing the new trend and "forgets" old behavior over time. The size of the window is an important parameter: If $w$ is too small, there will be many false positives; if it is too big, the network may not generalize well to novel sequences. The most recent of such systems (Debar et al. 1992) can predict the next command correctly around 80% of the time, and accept a command as predictable (among the three most likely next commands) 90% of the time.

One problem with the on-line approach is that most of the effort goes into predicting the order of commands. In many cases, the order does not matter much, but the distribution of commands that are used is revealing. A possibly effective approach could therefore be to collect statistics about the users' command usage over a period of time, such as a day, and try to recognize the distribution of commands as legitimate or anomalous off-line. This is the idea behind the NNID system.

## 3   THE NNID SYSTEM

The NNID anomaly intrusion detection system is based on identifying a legitimate user based on the distribution of commands she or he executes. This is justifiable because different users tend to exhibit different behavior, depending on their needs of the system. Some use the system to send and receive e-mail only, and do not require services such as programming and compilation. Some engage in all kinds of activities including editing, programming, e-mail, Web browsing, and so on. However, even two users that do the same thing may not use the same application program. For example, some may prefer the "vi" editor to "emacs", favor "pine" over "elm" as their mail utility program, or use "gcc" more often than "cc" to compile C programs. Also, the frequency with which a command is used varies from user to user. The set of commands used and their frequency, therefore, constitutes a 'print' of the user, reflecting the task performed and the choice of application programs, and it should be possible to identify the user based on this information.

It should be noted that this approach works even if some users have aliases set up as short-hands for long commands they use frequently, because the audit log records the actual commands executed by the system. Users' privacy is not violated, since the arguments to a command do not need to be recorded. That is, we may know that a user sends e-mail five times a day, but we do not need to know to whom the mail is addressed.

Building NNID for a particular computer system consists of the following three phases:

1. Collecting training data: Obtain the audit logs for each user for a period of several days. For each day and user, form a vector that represents how often the user executed each command.

| as | awk | bc | bibtex | calendar | cat | chmod | comsat | cp | cpp |
|---|---|---|---|---|---|---|---|---|---|
| cut | cvs | date | df | diff | du | dvips | egrep | elm | emacs |
| expr | fgrep | filter | find | finger | fmt | from | ftp | gcc | gdb |
| ghostview | gmake | grep | gs | gzip | hostname | id | ifconfig | ispell | last |
| ld | less | look | lpq | lpr | lprm | ls | machine | mail | make |
| man | mesg | metamail | mkdir | more | movemail | mpage | mt | mv | netscape |
| netstat | nm | objdump | perl | pgp | ping | ps | pwd | rcp | resize |
| rm | rsh | sed | sendmail | sh | sort | strip | stty | tail | tar |
| tcsh | tee | test | tgif | top | tput | tr | tty | uname | vacation |
| vi | virtex | w | wc | whereis | xbiff++ | xcalc | xdvi | xhost | xterm |

Table 1: **The 100 commands used to describe user behavior.** The number of times the user executed each of these commands during the day was recorded, mapped into a nonlinear scale of 11 intervals, and concatenated into a 100-dimensional input vector, representing the usage pattern for that user for that day.

2. Training: Train the neural network to identify the user based on these command distribution vectors.

3. Performance: Let the network identify the user for each new command distribution vector. If the network's suggestion is different from the actual user, or if the network does not have a clear suggestion, signal an anomaly.

The particular implementation of NNID and the environment where it was tested is described in the next section.

## 4   EXPERIMENTS

The NNID system was built and tested on a machine that serves a particular research group at the Department of Electrical and Computer Engineering at the University of Texas at Austin. This machine has 10 total users; some are regular users, with several other users logging in intermittently. This platform was chosen for three reasons:

1. The operating system (NetBSD) provides audit trail logging for accounting purposes and this option had been enabled on this system.

2. The number of users and the total number of commands executed per day are on an order of magnitude that is manageable. Thus, the feasibility of the approach could be tested with real-world data without getting into scalability issues.

3. The system is relatively unknown to outsiders and the users are all known to us, so that it is likely that the data collected on it consists of normal user behavior (free of intrusions).

Data was collected on this system for 12 days, resulting in 89 user-days. Instead of trying to optimize the selection of features (commands) for the input, we decided to simply use a set of 100 most common commands in the logs (listed in Table 1), and let the network figure out what information was important and what superfluous. Intelligent selection of features might improve the results some but the current approach is easy to implement and proves the point.

In order to introduce more overlap between input vectors, and therefore better generalization, the number of times a command was used was divided into intervals. There were 11 intervals, non-linearly spaced, so that the representation is more accurate at lower frequencies where it is most important. The first interval meant the command was never used; the second that it was used once or twice, and so on until the last interval where the command was used more than 500 times. The intervals were represented by values from 0.0 to 1.0 in 0.1 increments. These values, one for each command, were then concatenated into a 100-dimensional command distribution vector (also called user vector below) to be used as input to the neural network.

The standard three-layer backpropagation architecture was chosen for the neural network. The idea was to get results on the most standard and general architecture so that the feasibility of the approach could be demonstrated and the results would be easily replicable. More sophisticated architectures could be used and they would probably lead to slightly better results. The input layer consisted of 100 units, representing the user vector; the hidden layer had 30 units and the output layer 10 units, one for each user. The network was implemented in the PlaNet Neural Network simulator (Miyata 1991).

## 5  RESULTS

To avoid overtraining, several training sessions were run prior to the actual experiments to see how many training cycles would give the highest performance. The network was trained on 8 randomly chosen days of data (65 user vectors), and its performance was tested on the remaining 4 days (24 vectors) after epochs 30, 50, 100, 200, and 300, of which 100 gave the best performance. Four splits of the data into training and testing sets were created by randomly picking 8 days for training. The resulting four networks were tested in two tasks:

1. Identifying the user vectors of the remaining 4 days. If the activation of the output unit representing the correct user was higher than those of all other units, and also higher than 0.5, the identification was counted as correct. Otherwise, a false positive was counted.

2. Identifying 100 randomly-generated user vectors. If all output units had an activation less than 0.5, the network was taken to correctly identify the vector as an anomaly (i.e. not any of the known users in the system). Otherwise, the most highly active output unit identifies the network's suggestion. Since all intrusions occur under one of the 10 user accounts, there is a 1/10 chance that the suggestion would accidentally match the compromised user account and the intrusion would not be detected. Therefore, 1/10 of all such cases were counted as false negatives.

The second test is a suggestive measure of the accuracy of the system. It is not possible to come up with vectors that would represent a good sampling of actual intrusions; the idea here was to generate vectors where the values for each command were randomly drawn from the distribution of values for that command in the entire data set. In other words, the random test vectors had the same first-order statistics as the legitimate user vectors, but had no higher-order correlations. Therefore they constitute a neutral but realistic sample of unusual behavior.

All four splits led to similar results. On average, the networks rejected 63% of the random user vectors, leading to an anomaly detection rate of 96%. They correctly identified the legitimate user vectors 93% of the time, giving a false alarm rate of 7%.

Figure 1 shows the output of the network for one of the splits. Out of 24 legitimate user vectors, the network identified 22. Most of the time the correct output unit is very highly activated, indicating high certainty of identification. However, the activation of the highest unit was below 0.5 for two of the inputs, resulting in a false alarm.

Interestingly, in all false alarms in all splits, the falsely-accused user was always the same. A closer look at the data set revealed that there were only 3 days of data on this user. He used the system very infrequently, and the network could not learn a proper profile for him. While it would be easy to fix this problem by collecting more data in this case, we believe this is a problem that would be difficult to rule out in general. No matter how much data one collects, there may still not be enough for some extremely infrequent user. Therefore, we believe the results obtained in this rather small data set give a realistic picture of the performance of the NNID system.

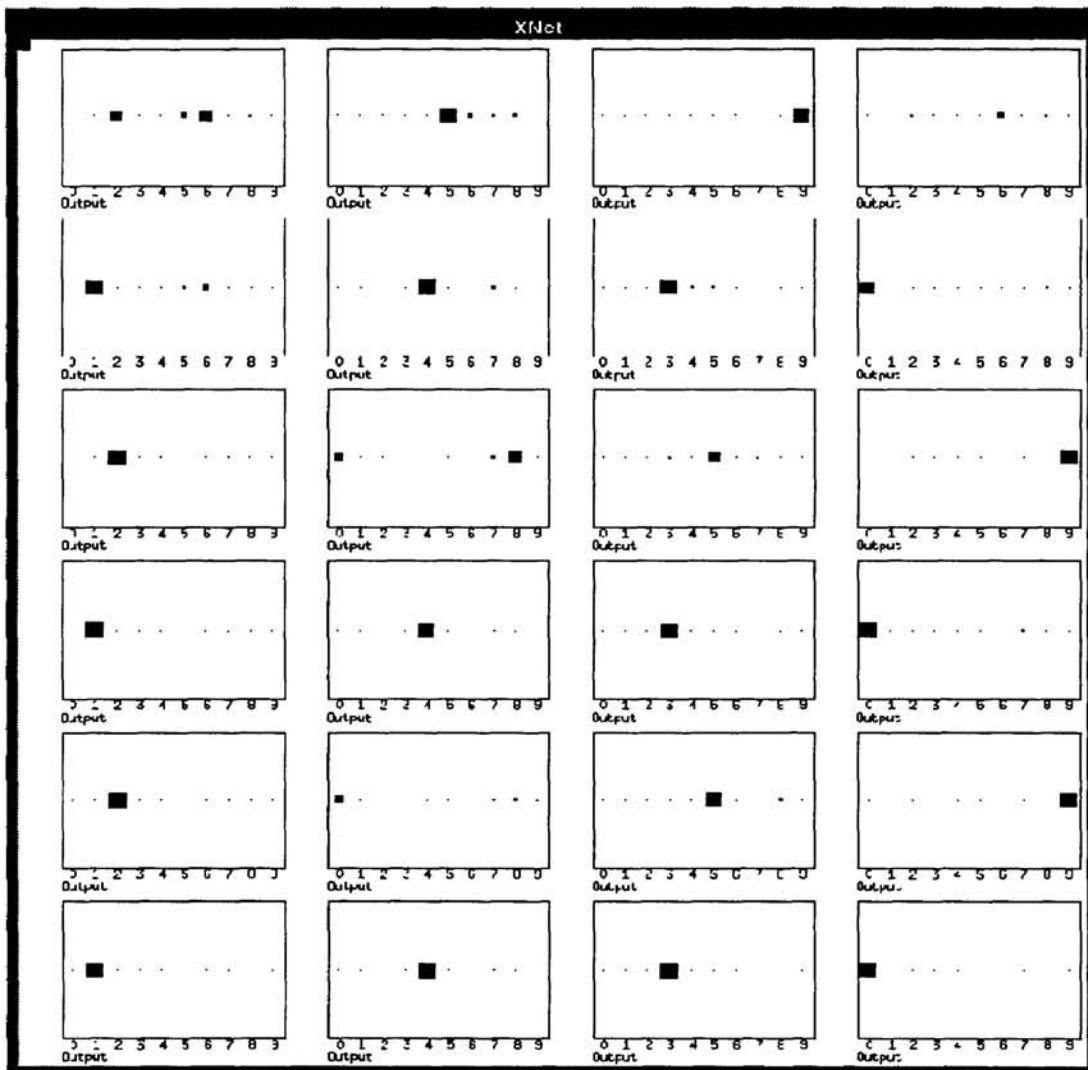

Figure 1: **User identification with the NNID Network.** The output layer of NNID is shown for each of the 24 test vectors in one of the 4 splits tested. The output units are lined up from left to right, and their activations are represented by the size of the squares. In this split there were two false alarms: one is displayed in the top right with activation 0.01, and one in the second row from the bottom, second column from the left with 0.35. All the other test vectors are identified correctly with activation higher than 0.5.

## 6   DISCUSSION AND FUTURE WORK

An important question is, how well does the performance of NNID scale with the number of users? Although there are many computer systems that have no more than a dozen users, most intrusions occur in larger systems with hundreds of users. With more users, the network would have to make finer distinctions, and it would be difficult to maintain the same low level of false alarms. However, the rate of detecting anomalies may not change much, as long as the network can learn the user patterns well. Any activity that differs from the user's normal behavior would still be detected as an anomaly.

Training the network to represent many more users may take longer and require a larger network, but it should be possible because the user profiles share a lot of common structure, and neural networks in general are good at learning such data. Optimizing the set of commands included in the user vector, and the size of the value intervals, might also have a large impact on performance. It would be interesting to determine the curve of performance

versus the number of users, and also see how the size of the input vector and the granularity of the value intervals affect that curve. This is the most important direction of future work.

Another important issue is, how much does a user's behavior change over time? If behavior changes dramatically, NNID must be recalibrated often or the number of false positives would increase. Fortunately such retraining is easy to do. Since NNID parses daily activity of each user into a user-vector, the user profile can be updated daily. NNID could then be retrained periodically. In the current system it takes only about 90 seconds and would not be a great burden on the system.

## 7 CONCLUSION

Experimental evaluation on real-world data shows that NNID can learn to identify users simply by what commands they use and how often, and such an identification can be used to detect intrusions in a network computer system. The order of commands does not need to be taken into account. NNID is easy to train and inexpensive to run because it operates off-line on daily logs. As long as real-time detection is not required, NNID constitutes a promising, practical approach to anomaly intrusion detection.

### Acknowledgements

Special thanks to Mike Dahlin and Tom Ziaja for feedback on an earlier version of this paper, and to Jim Bednar for help with the PlaNet simulator. This research was supported in part by DOD-ARPA contract F30602-96-1-0313, NSF grant IRI-9504317, and the Texas Higher Education Coordinating board grant ARP-444.

## Footnotes

*Currently: MCI Communications Corp., 9001 N. IH 35, Austin, TX 78753; jake.ryan@mci.com.

## References

Debar, H., Becker, M., and Siboni, D. (1992). A neural network component for an intrusion detection system. In *Proceedings of the 1992 IEEE Computer Society Symposium on Research in Computer Security and Privacy*, 240–250.

Denning, D. E. (1987). An intrusion detection model. *IEEE Transactions on Software Engineering*, SE-13:222–232.

Fox, K. L., Henning, R. R., Reed, J. H., and Simonian, R. (1990). A neural network approach towards intrusion detection. In *Proceedings of the 13th National Computer Security Conference*, 125–134.

Frank, J. (1994). Artificial intelligence and intrusion detection: Current and future directions. In *Proceedings of the National 17th Computer Security Conference*.

Garvey, T. D., and Lunt, T. F. (1991). Model-based intrusion detection. In *Proceedings of the 14th National Computer Security Conference*.

Miyata, Y. (1991). *A User's Guide to PlaNet Version 5.6 – A Tool for Constructing, Running, and Looking in to a PDP Network*. Computer Science Department, University of Colorado, Boulder, Boulder, CO.

Mukherjee, B., Heberlein, L. T., and Levitt, K. N. (1994). Network intrusion detection. *IEEE Network*, 26–41.

Porras, P. A., Ilgun, K., and Kemmerer, R. A. (1995). State transition analysis: A rule-based intrusion detection approach. *IEEE Transactions on Software Engineering*, SE-21:181–199.

Teng, H. S., Chen, K., and Lu, S. C. (1990). Adaptive real-time anomaly detection using inductively generated sequential patterns. In *Proceedings of the 1990 IEEE Symposium on Research in Computer Security and Privacy*, 278–284.